# Incremental Gaussian Processes

**Joaquin Quiñonero-Candela**
Informatics and Mathematical Modelling
Technical University of Denmark
DK-2800 Lyngby, Denmark
jqc@imm.dtu.dk

**Ole Winther**
Informatics and Mathematical Modelling
Technical University of Denmark
DK-2800 Lyngby, Denmark
owi@imm.dtu.dk

## Abstract

In this paper, we consider Tipping's relevance vector machine (RVM) [1] and formalize an incremental training strategy as a variant of the expectation-maximization (EM) algorithm that we call Subspace EM (SSEM). Working with a subset of active basis functions, the sparsity of the RVM solution will ensure that the number of basis functions and thereby the computational complexity is kept low. We also introduce a mean field approach to the intractable classification model that is expected to give a very good approximation to exact Bayesian inference and contains the Laplace approximation as a special case. We test the algorithms on two large data sets with $\mathcal{O}(10^3 - 10^4)$ examples. The results indicate that Bayesian learning of large data sets, e.g. the MNIST database is realistic.

## 1 Introduction

Tipping's relevance vector machine (RVM) both achieves a sparse solution like the support vector machine (SVM) [2, 3] and the probabilistic predictions of Bayesian kernel machines based upon a Gaussian process (GP) priors over functions [4, 5, 6, 7, 8]. Sparsity is interesting both with respect to fast training and predictions and ease of interpretation of the solution. Probabilistic predictions are desirable because inference is most naturally formulated in terms of probability theory, i.e. we can manipulate probabilities through Bayes theorem, reject uncertain predictions, etc.

It seems that Tipping's relevance vector machine takes the best of both worlds. It is a GP with a covariance matrix spanned by a small number of basis functions making the computational expensive matrix inversion operation go from $\mathcal{O}(N^3)$, where $N$ is the number of training examples to $\mathcal{O}(M^2N)$ ($M$ being the number of basis functions). Simulation studies have shown very sparse solutions $M \ll N$ and good test performance [1]. However, starting the RVM learning with as many basis functions as examples, i.e. one basis function in each training input point, leads to the same complexity as for Gaussian processes (GP) since in the initial step no basis functions are removed. That lead Tipping to suggest in an appendix in Ref. [1] an incremental learning strategy that starts with only a single basis function and adds basis functions along the iterations, and to formalize it very recently [9]. The total number of basis functions is kept low because basis functions are also removed. In this paper we formalize this strategy using straightforward expectation-maximization (EM) [10] arguments to prove that the scheme is the guaranteed convergence to a local

maximum of the likelihood of the model parameters.

Reducing the computational burden of Bayesian kernel learning is a subject of current interest. This can be achieved by numerical approximations to matrix inversion [11] and suboptimal projections onto finite subspaces of basis functions without having an explicit parametric form of such basis functions [12, 13]. Using mixtures of GPs [14, 15] to make the kernel function input dependent is also a promising technique. None of the Bayesian methods can currently compete in terms of speed with the efficient SVM optimization schemes that have been developed, see e.g. [3].

The rest of the paper is organized as follows: In section 2 we present the extended linear models in a Bayesian perspective, the regression model and the standard EM approach. In section 3, a variation of the EM algorithm, that we call the *Subspace EM* (SSEM) is introduced that works well with sparse solution models. In section 4, we present the second main contribution of the paper: a mean field approach to RVM classification. Section 5 gives results for the Mackey-Glass time-series and preliminary results on the MNIST hand-written digits database. We conclude in section 6.

## 2   Regression

An extended linear model is built by transforming the input space by an arbitrary set of *basis functions* $\phi_j : R^D \rightarrow R$ that performs a non-linear transformation of the $D$-dimensional input space. A linear model is applied to the transformed space whose dimension is equal to the number of basis functions $M$:

$$y(\mathbf{x}_i) = \sum_{j=1}^{M} \omega_j \, \phi_j(\mathbf{x}_i) = \mathbf{\Phi}(\mathbf{x}_i) \cdot \boldsymbol{\omega} \tag{1}$$

where $\mathbf{\Phi}(\mathbf{x}_i) \equiv [\phi_1(\mathbf{x}_i), \dots, \phi_M(\mathbf{x}_i)]$ denotes the $i$th row of the *design matrix* $\mathbf{\Phi}$ and $\boldsymbol{\omega} = (\omega_1, \dots, \omega_N)^T$ is the *weights* vector. The output of the model is thus a linear superposition of completely general basis functions. While it is possible to optimize the parameters of the basis functions for the problem at hand [1, 16], we will in this paper assume that they are given.

The simplest possible regression learning scenario can be described as follows: a set of $N$ input-target training pairs $\{x_i, t_i\}_{i=1}^{N}$ are assumed to be independent and contaminated with Gaussian noise of variance $\sigma^2$. The likelihood of the parameters $\boldsymbol{\omega}$ is given by

$$p(\mathbf{t}|\boldsymbol{\omega}, \sigma^2) = \left(2\pi\sigma^2\right)^{-N/2} \exp\left(-\frac{1}{2\sigma^2} \|\mathbf{t} - \mathbf{\Phi}\boldsymbol{\omega}\|^2\right) \tag{2}$$

where $\mathbf{t} = (t_1, \dots, t_N)^T$ is the target vector. Regularization is introduced in Bayesian learning by means of a prior distribution over the weights. In general, the implied prior over functions is a very complicated distribution. However, choosing a Gaussian prior on the weights the prior over functions also becomes Gaussian, i.e. a Gaussian process. For the specific choice of a factorized distribution with variance $\alpha_j^{-1}$:

$$p(\omega_j|\alpha_j) = \sqrt{\frac{\alpha_j}{2\pi}} \exp\left(-\frac{1}{2}\alpha_j\,\omega_j^2\right) \tag{3}$$

the prior over functions $p(\mathbf{y}|\boldsymbol{\alpha})$ is $\mathcal{N}(0, \mathbf{\Phi}\mathbf{A}^{-1}\mathbf{\Phi}^T)$, i.e. a Gaussian process with covariance function given by

$$\text{Cov}(\mathbf{x}_i, \mathbf{x}_j) = \sum_{k=1}^{M} \frac{1}{\alpha_k}\phi_k(x_i)\phi_k(x_j) \tag{4}$$

where $\boldsymbol{\alpha} = (\alpha_0, \ldots, \alpha_N)^T$ and $\mathbf{A} = \mathrm{diag}(\alpha_0, \ldots, \alpha_N)$. We can now see how sparseness in terms of the basis vectors may arise: if $\alpha_k^{-1} = 0$ the $k$th basis vector $\boldsymbol{\Phi}_k \equiv [\phi_k(\mathbf{x}_1), \ldots, \phi_k(\mathbf{x}_N)]^T$, i.e. the $k$th column in the design matrix, will not contribute to the model. Associating a basis function with each input point may thus lead to a model with a sparse representations in the inputs, i.e. the solution is only spanned by a subset of all input points. This is exactly the idea behind the relevance vector machine, introduced by Tipping [17]. We will see in the following how this also leads to a lower computational complexity than using a regular Gaussian process kernel.

The posterior distribution over the weights–obtained through Bayes rule–is a Gaussian distribution

$$p(\boldsymbol{\omega}|\mathbf{t}, \boldsymbol{\alpha}, \sigma^2) = \frac{p(\mathbf{t}|\boldsymbol{\omega}, \sigma^2)p(\boldsymbol{\omega}|\boldsymbol{\alpha})}{p(\mathbf{t}|\boldsymbol{\alpha}, \sigma^2)} = \mathcal{N}(\boldsymbol{\omega}|\boldsymbol{\mu}, \boldsymbol{\Sigma}) \tag{5}$$

where $\mathcal{N}(\mathbf{t}|\boldsymbol{\mu}, \boldsymbol{\Sigma})$ is a Gaussian distribution with mean $\boldsymbol{\mu}$ and covariance $\boldsymbol{\Sigma}$ evaluated at $\mathbf{t}$. The mean and covariance are given by

$$\boldsymbol{\mu} = \sigma^{-2}\boldsymbol{\Sigma}\boldsymbol{\Phi}^T\mathbf{t} \tag{6}$$
$$\boldsymbol{\Sigma} = (\sigma^{-2}\boldsymbol{\Phi}^T\boldsymbol{\Phi} + \mathbf{A})^{-1} \tag{7}$$

The uncertainty about the optimal value of the weights captured by the posterior distribution (5) can be used to build probabilistic predictions. Given a new input $\mathbf{x}_*$, the model gives a Gaussian *predictive distribution* of the corresponding target $t_*$

$$p(t_*|\mathbf{x}_*, \boldsymbol{\alpha}, \sigma^2) = \int p(t_*|\mathbf{x}_*, \boldsymbol{\omega}, \sigma^2)\, p(\boldsymbol{\omega}|\mathbf{t}, \boldsymbol{\alpha}, \sigma^2)\, d\boldsymbol{\omega} = \mathcal{N}(t_*|y_*, \sigma_*^2) \tag{8}$$

where

$$y_* = \boldsymbol{\Phi}(\mathbf{x}_*) \cdot \boldsymbol{\mu} \tag{9}$$
$$\sigma_*^2 = \sigma^2 + \boldsymbol{\Phi}(\mathbf{x}_*) \cdot \boldsymbol{\Sigma} \cdot \boldsymbol{\Phi}(\mathbf{x}_*)^T \tag{10}$$

For regression it is natural to use $y_*$ and $\sigma_*$ as the prediction and the error bar on the prediction respectively. The computational complexity of making predictions is thus $\mathcal{O}(M^2 P + M^3 + M^2 N)$, where $M$ is the number of selected basis functions (RVs) and $P$ is the number of predictions. The two last terms come from the computation of $\boldsymbol{\Sigma}$ in eq. (7).

The likelihood distribution over the training targets (2) can be "marginalized" with respect to the weights to obtain the *marginal likelihood*, which is also a Gaussian distribution

$$p(\mathbf{t}|\boldsymbol{\alpha}, \sigma^2) = \int p(\mathbf{t}|\boldsymbol{\omega}, \sigma^2)\, p(\boldsymbol{\omega}|\boldsymbol{\alpha})\, d\boldsymbol{\omega} = \mathcal{N}(\mathbf{t}|0, \sigma^2\mathbf{I} + \boldsymbol{\Phi}\mathbf{A}^{-1}\boldsymbol{\Phi}^T)\,. \tag{11}$$

Estimating the hyperparameters $\{\alpha_j\}$ and the noise $\sigma^2$ can be achieved by maximizing (11). This is naturally carried out in the framework of the expectation-maximization (EM) algorithm since the sufficient statistics of the weights (that act as hidden variables) are available for this type of model. In other cases e.g. for adapting the length scale of the kernel [4], gradient methods have to be used. For regression, the E-step is exact (the lower bound on the marginal likelihood is made equal to the marginal likelihood) and consists in estimating the mean and variance (6) and (7) of the posterior distribution of the weights (5). For classification, the E-step will be approximate. In this paper we present a mean field approach for obtaining the sufficient statistics.

The M-step corresponds to maximizing the expectation of the log marginal likelihood with respect to the posterior, with respect to $\sigma^2$ and $\boldsymbol{\alpha}$, which gives the following update rules:
$\alpha_j^{new} = \frac{1}{\langle \omega_j^2 \rangle_{p(\boldsymbol{\omega}|\mathbf{t}, \boldsymbol{\alpha}, \sigma^2)}} = \frac{1}{\mu_j^2 + \Sigma_{jj}}$, and $(\sigma^2)^{new} = \frac{1}{N}(||\mathbf{t} - \boldsymbol{\Phi}\,\mu||^2 + (\sigma^2)^{old}\sum_j \gamma_j)$,

where the quantity $\gamma_j \equiv 1 - \alpha_j \Sigma_{jj}$ is a measure of how "well-determined" each weight $\omega_j$ is by the data [18, 1]. One can obtain a different update rule that gives faster convergence. Although it is suboptimal in the EM sense, we have never observed it decrease the lower bound on the marginal log-likelihood. The rule, derived in [1], is obtained by differentiation of (11) and by an arbitrary choice of independent terms as is done by [18]. It makes use of the terms $\{\gamma_j\}$:

$$\alpha_j^{new} = \frac{\gamma_j}{\mu_j^2} \qquad\qquad (\sigma^2)^{new} = \frac{||\mathbf{t} - \mathbf{\Phi}\,\mu||^2}{N - \sum_j \gamma_j} \; . \tag{12}$$

In the optimization process many $\alpha_j$ grow to infinity, which effectively deletes the corresponding weight and basis function. Note that the EM update and the Mackay update for $\alpha_j$ only implicitly depend upon the likelihood. This means that it is also valid for the classification model we shall consider below.

A serious limitation of the EM algorithm and variants for problems of this type is that the complexity of computing the covariance of the weights (7) in the E-step is $O(M^3 + M^2 N)$. At least in the first iteration where no basis functions have been deleted $M = N$ and we are facing the same kind of complexity explosion that limits the applicability of Gaussian processes to large training set. This has lead Tipping [1] to consider a constructive or incremental training paradigm where one basis function is added before each E-step and since basis functions are removed in the M-step, it turns out in practice that the total number of basis functions and the complexity remain low [9]. In the following section we introduce a new algorithm that formalizes this procedure that can be proven to increase the marginal likelihood in each step.

## 3   Subspace EM

We introduce an incremental approach to the EM algorithm, the *Subspace EM* (SSEM), that can be directly applied to training models like the RVM that rely on a linear superposition of completely general basis functions, both for classification and for regression. Instead of starting with a full model, i.e. where all the basis functions are present with finite $\alpha$ values, we start with a fully pruned model with all $\alpha_j$ set to infinity. Effectively, we start with no model. The model is grown by iteratively including some $\alpha_j$ previously set to infinity to the *active set* of $\alpha$'s. The active set at iteration $n$, $R_n$, contains the indices of the basis vectors with $\alpha$ less than infinity:

$$\begin{aligned} R_1 &= 1 \\ R_n &= \{i \mid i \in R_{n-1} \wedge \alpha_i \leq \mathrm{L}\} \cup \{n\} \end{aligned} \tag{13}$$

where L is a finite very large number arbitrarily defined. Observe that $R_n$ contains at most one more element (index) than $R_{n-1}$. If some of the $\alpha$'s indexed by $R_{n-1}$ happen to reach L at the $n$-th step, $R_n$ can contain less elements than $R_{n-1}$. In figure 1 we give a schematic description of the SSEM algorithm.

At iteration $n$ the E-step is taken only in the subspace spanned by the weights whose indexes are in $R_n$. This helps reducing the computational complexity of the M-step to $O(M^3)$, where $M$ is the number of relevance vectors.

Since the initial value of $\alpha_j$ is infinity for all $j$, for regression the E-step yields always an equality between the log marginal likelihood and its lower bound. At any step $n$, the posterior can be exactly projected on to the space spanned by the weights $\omega_j$ such that $j \in R_n$, because the $\alpha_k = \infty$ for all $k$ not in $R_n$. Hence in the regression case, the SSEM never decreases the log marginal likelihood. Figure 2 illustrates the convergence process of the SSEM algorithm compared to that of the EM algorithm for regression.

```
1.  Set αⱼ = L for all j. (L is a very large number) Set n = 1
2.  Update the set of active indexes Rₙ
3.  Perform an E-step in subspace ωⱼ such that j ∈ Rₙ
4.  Perform the M-step for all αⱼ such that j ∈ Rₙ
5.  If visited all basis functions, end, else go to 2.
```

Figure 1: Schematics of the SSEM algorithm.

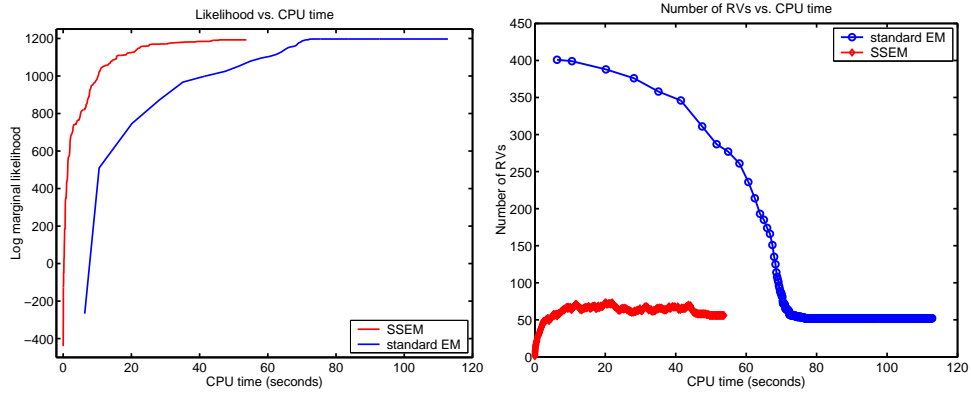

Figure 2: Training on 400 samples of the Mackey-Glass time series, testing on 2000 cases. Log marginal likelihood as a function of the elapsed CPU time (left) and corresponding number of relevance vectors (right) for both SSEM and EM.

We perform one EM step for each time a new basis function is added to the active set. Once all the examples have been visited, we switch to the batch EM algorithm *on the active set* until some convergence criteria has been satisfied, for example until the relative increase in the likelihood is smaller than a certain threshold. In practice some $50$ of these batch EM iterations are enough.

## 4  Classification

Unlike the model discussed above, analytical inference is not possible for classification models. Here, we will discuss the adaptive TAP mean field approach–initially proposed for Gaussian processes [8]–that are readily translated to RVMs. The mean field approach has the appealing features that it retains the computational efficiency of RVMs, is exact for the regression and reduces to the Laplace approximation in the limit where all the variability comes from the prior distribution.

We consider binary $t = \pm 1$ classification using the *probit* likelihood with 'input' noise $\sigma^2$

$$p(t|y(\mathbf{x})) = \mathrm{erf}\left(t\frac{y(\mathbf{x})}{\sigma}\right) \ , \tag{14}$$

where $Dz \equiv e^{-z^2/2}dz/\sqrt{2\pi}$ and $\mathrm{erf}(x) \equiv \int_{-\infty}^{x} Dz$ is an error function (or cumulative Gaussian distribution). The advantage of using this sigmoid rather than the commonly used 0/1-logistic is that we under the mean field approximation can derive an analytical expression for the predictive distribution $p(t_*|\mathbf{x}_*, \mathbf{t}) = \int p(t_*|y)p(y|\mathbf{x}_*, \mathbf{t})dy$ needed for making Bayesian predictions. Both a variational and the advanced mean field approach–used here–make a Gaussian approximation for $p(y|\mathbf{x}_*, \mathbf{t})$ [8] with mean and variance given by regression results $y_*$ and $\sigma_*^2 - \hat{\sigma}^2$, and $y_*$ and $\sigma_*^2$ given by eqs. (9) and (10). This leads

to the following approximation for the predictive distribution

$$p(t_*|\mathbf{x}_*, \mathbf{t}) = \int \mathrm{erf}\left(t_* \frac{y}{\sigma}\right) p(y|\mathbf{x}_*, \mathbf{t}) \, dy = \mathrm{erf}\left(t_* \frac{y_*}{\sigma_*}\right) \ . \tag{15}$$

However, the mean and covariance of the weights are no longer found by analytical expressions, but has to be obtained from a set of non-linear mean field equations that also follow from equivalent assumptions of Gaussianity for the training set outputs $y(\mathbf{x}_i)$ in averages over reduced (or cavity) posterior averages.

In the following, we will only state the results which follows from combining the RVM Gaussian process kernel (4) with the results of [8]. The sufficient statistics of the weights are written in terms of a set of $\mathcal{O}(N)$ mean field parameters

$$\boldsymbol{\mu} = \mathbf{A}^{-1}\boldsymbol{\Phi}^T\boldsymbol{\tau} \tag{16}$$

$$\boldsymbol{\Sigma} = \left(\mathbf{A} + \boldsymbol{\Phi}^T\boldsymbol{\Omega}\boldsymbol{\Phi}\right)^{-1} \tag{17}$$

where $\tau_i \equiv \frac{\partial}{\partial y_i^c} \ln Z(y_i^c, V_i^c + \sigma^2)$ and

$$Z(y_i^c, V_i^c + \sigma^2) \equiv \int p(t_i|y_i^c + z\sqrt{V_i^c + \sigma^2}) \, Dz = \mathrm{erf}\left(t_i \frac{y_i^c}{\sqrt{V_i^c + \sigma^2}}\right) \ . \tag{18}$$

The last equality holds for the likelihood eq. (14) and $y_i^c$ and $V_i^c$ are the mean and variance of the so called cavity field. The mean value is $y_i^c = \boldsymbol{\Phi}(\mathbf{x}_i) \cdot \boldsymbol{\mu} - V_i^c \tau_i$. The distinction between the different approximation schemes is solely in the variance $V_i^c$: $V_i^c = 0$ is the Laplace approximation, $V_i^c = \left[\boldsymbol{\Phi}\mathbf{A}^{-1}\boldsymbol{\Phi}^T\right]_{ii}$ is the so called naive mean field theory and an improved estimate is available from the adaptive TAP mean field theory [8]. Lastly, the diagonal matrix $\boldsymbol{\Omega}$ is the equivalent of the noise variance in the regression model (compare eqs. (17) and (7) and is given by $\Omega_i = -\frac{\partial \tau_i}{\partial y_i^c}/(1 + V_i^c \frac{\partial \tau_i}{\partial y_i^c})$. This set of non-linear equations are readily solved (i.e. fast and stable) by making Newton-Raphson updates in $\boldsymbol{\mu}$ treating the remaining quantities as help variables:

$$\Delta\boldsymbol{\mu} = (\mathbf{I} + \mathbf{A}^{-1}\boldsymbol{\Phi}^T\boldsymbol{\Omega}\boldsymbol{\Phi})^{-1}(\mathbf{A}^{-1}\boldsymbol{\Phi}^T\boldsymbol{\tau} - \boldsymbol{\mu}) = \boldsymbol{\Sigma}(\boldsymbol{\Phi}^T\boldsymbol{\tau} - \mathbf{A}\boldsymbol{\mu}) \tag{19}$$

The computational complexity of the E-step for classification is augmented with respect to the regression case by the fact that it is necessary to construct and invert a $M \times M$ matrix usually many times (typically 20), once for each step of the iterative Newton method.

## 5 Simulations

We illustrate the performance of the SSEM for regression on the Mackey-Glass chaotic time series, which is well-known for its strong non-linearity. In [16] we showed that the RVM has an order of magnitude superior performance than carefully tuned neural networks for time series prediction on the Mackey-Glass series. The inputs are formed by $L = 16$ samples spaced 6 periods from each other $\mathbf{x}_k = [z(k-6), z(k-12), \ldots, z(k-6L)]$ and the targets are chosen to be $t_k = z(k)$ to perform six steps ahead prediction (see [19] for details). We use Gaussian basis functions of fixed variance $\nu^2 = 10$. The test set comprises 5804 examples.

We perform prediction experiments for different sizes of the training set. We perform in each case 10 repetitions with different partitions of the data sets into training and test. We compare the test error, the number of RVs selected and the computer time needed for the batch and the SSEM method. We present the results obtained with the growth method relative to the results obtained with the batch method in figure 3. As expected, the relative

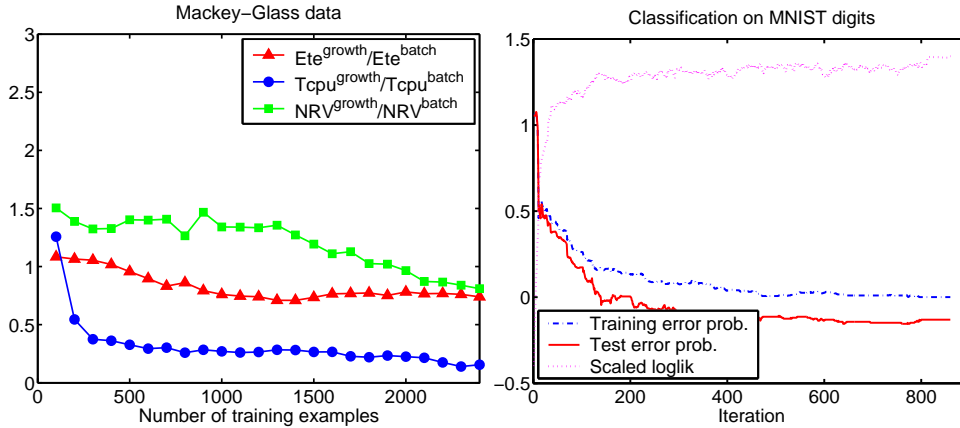

Figure 3: Left: Regression, mean values over 10 repetitions of relative test error, number of RVs and computer time for the Mackey-Glass data, up to 2400 training examples and 5804 test examples. Right: Classification, Log marginal likelihood, test and training errors while training on one class against all the others, 60000 training and 10000 test examples.

computer time of the growth method compared with the batch method decreases with size of the training set. For a few thousand examples the SSEM method is an order of magnitude faster than the batch method. The batch method proved only to be faster for $100$ training examples, and could not be used with data sets of thousands of examples on the machine on which we run the experiments because of its high memory requirements. This is the reason why we only ran the comparison for up to $2400$ training example for the Mackey-Glass data set.

Our experiments for classification are at the time of sending this paper to press very premature: we choose a very large data set, the MNIST database of handwritten digits [20], with $60000$ training and $10000$ test images. The images are of size $28 \times 28$ pixels. We use PCA to project them down to $16$ dimensional vectors. We only perform a preliminary experiment consisting of a one against all binary classification problem to illustrate that Bayesian approaches to classification can be used on very large data sets with the SSEM algorithm. We train on $13484$ examples (the $6742$ *one*'s and another $6742$ random non-*one* digits selected at random from the rest) and we use $800$ basis functions for both the batch and Subspace EM. In figure 3 we show the convergence of the SSEM in terms of the log marginal likelihood and the training and test probabilities of error. The test probability of error we obtain is $0.74$ percent with the SSEM algorithm and $0.66$ percent with the batch EM. Under the same conditions the SSEM needed $55$ minutes to do the job, while the batch EM needed $186$ minutes. The SSEM gives a machine with $28$ basis functions and the batch EM one with $31$ basis functions.

# 6   Conclusion

We have presented a new approach to Bayesian training of linear models, based on a subspace extension of the EM algorithm that we call *Subspace EM* (SSEM). The new method iteratively builds models from a potentially big library of basis functions. It is especially well-suited for models that are constructed such that they yield a sparse solution, i.e. the solution is spanned by small number $M$ of basis functions, which is much smaller than $N$, the number of examples. A prime example of this is Tipping's relevance vector machine that typically produces solutions that are sparser than those of support vector machines. With

the SSEM algorithm the computational complexity and memory requirement decrease from $\mathcal{O}(N^3)$ and $\mathcal{O}(N^2)$ to $\mathcal{O}(M^2 N)$ (somewhat higher for classification) and $\mathcal{O}(NM)$. For classification, we have presented a mean field approach that is expected to be a very good approximation to the exact inference and contains the widely used Laplace approximation as an extreme case. We have applied the SSEM algorithm to both a large regression and a large classification data sets. Although preliminary for the latter, we believe that the results demonstrate that Bayesian learning is possible for very large data sets. Similar methods should also be applicable beyond supervised learning.

**Acknowledgments** JQC is funded by the EU *Multi-Agent Control* Research Training Network - EC TMR grant HPRNCT-1999-00107. We thank Lars Kai Hansen for very useful discussions.

# References

[1] Michael E. Tipping, "Sparse bayesian learning and the relevance vector machine," *Journal of Machine Learning Research*, vol. 1, pp. 211–244, 2001.

[2] Vladimir N. Vapnik, *Statistical Learning Theory*, Wiley, New York, 1998.

[3] Bernhard Schölkopf and Alex J. Smola, *Learning with Kernels*, MIT Press, Cambridge, 2002.

[4] Carl E. Rasmussen, *Evaluation of Gaussian Processes and Other Methods for Non-linear Regression*, Ph.D. thesis, Dept. of Computer Science, University of Toronto, 1996.

[5] Chris K. I. Williams and Carl E. Rasmussen, "Gaussian Proceses for Regression," in *Advances in Neural Information Processing Systems*, 1996, number 8, pp. 514–520.

[6] D. J. C. Mackay, "Gaussian Processes: A replacement for supervised Neural Networks?," Tech. Rep., Cavendish Laboratory, Cambridge University, 1997, Notes for a tutorial at NIPS 1997.

[7] Radford M. Neal, *Bayesian Learning for Neural Networks*, Springer, New York, 1996.

[8] Manfred Opper and Ole Winther, "Gaussian processes for classification: Mean field algorithms," *Neural Computation*, vol. 12, pp. 2655–2684, 2000.

[9] Michael Tipping and Anita Faul, "Fast marginal likelihood maximisation for sparse bayesian models," in *International Workshop on Artificial Intelligence and Statistics*, 2003.

[10] N. M. Dempster, A.P. Laird, and D. B. Rubin, "Maximum likelihood from incomplete data via the EM algorithm," *J. R. Statist. Soc. B*, vol. 39, pp. 185–197, 1977.

[11] Chris Williams and Mathias Seeger, "Using the Nyström method to speed up kernel machines," in *Advances in Neural Information Processing Systems*, 2001, number 13, pp. 682–688.

[12] Alex J. Smola and Peter L. Bartlett, "Sparse greedy gaussian process regression," in *Advances in Neural Information Processing Systems*, 2001, number 13, pp. 619–625.

[13] Lehel Csató and Manfred Opper, "Sparse representation for gaussian process models," in *Advances in Neural Information Processing Systems*, 2001, number 13, pp. 444–450.

[14] Volker Tresp, "Mixtures of gaussian processes," in *Advances in Neural Information Processing Systems*, 2000, number 12, pp. 654–660.

[15] Carl E. Rasmussen and Zoubin Ghahramani, "Infinite mixtures of gaussian process experts," in *Advances in Neural Information Processing Systems*, 2002, number 14.

[16] Joaquin Quiñonero-Candela and Lars Kai Hansen, "Time series prediction based on the relevance vector machine with adaptive kernels," in *International Conference on Acoustics, Speech, and Signal Processing (ICASSP)*, 2002.

[17] Michael E. Tipping, "The relevance vector machine," in *Advances in Neural Information Processing Systems*, 2000, number 12, pp. 652–658.

[18] David J. C. MacKay, "Bayesian interpolation," *Neural Computation*, vol. 4, no. 3, pp. 415–447, 1992.

[19] Claus Svarer, Lars K. Hansen, Jan Larsen, and Carl E. Rasmussen, "Designer networks for time series processing," in *IEEE NNSP Workshop*, 1993, pp. 78–87.

[20] Y. LeCun, L. Bottou, Y. Bengio, and P. Haffner, "Gradient-based learning applied to document recognition," in *Poceedings of the IEEE*, 1998, vol. 86, pp. 2278–2324.
